# THE BOLTZMANN PERCEPTRON NETWORK:
# A MULTI-LAYERED FEED-FORWARD NETWORK
# EQUIVALENT TO THE BOLTZMANN MACHINE

Eyal Yair and Allen Gersho

Center for Information Processing Research
Department of Electrical & Computer Engineering
University of California, Santa Barbara, CA 93106

## ABSTRACT

The concept of the stochastic Boltzmann machine (BM) is attractive for decision making and pattern classification purposes since the probability of attaining the network states is a function of the network energy. Hence, the probability of attaining particular energy minima may be associated with the probabilities of making certain decisions (or classifications). However, because of its stochastic nature, the complexity of the BM is fairly high and therefore such networks are not very likely to be used in practice. In this paper we suggest a way to alleviate this drawback by converting the stochastic BM into a deterministic network which we call the Boltzmann Perceptron Network (BPN). The BPN is functionally equivalent to the BM but has a feed-forward structure and low complexity. No annealing is required. The conditions under which such a conversion is feasible are given. A learning algorithm for the BPN based on the conjugate gradient method is also provided which is somewhat akin to the backpropagation algorithm.

## INTRODUCTION

In decision-making applications, it is desirable to have a network which computes the probabilities of deciding upon each of $M$ possible propositions for any given input pattern. In principle, the Boltzmann machine (BM) (Hinton, Sejnowski and Ackley, 1984) can provide such a capability. The network is composed of a set of binary units connected through symmetric connection links. The units are randomly and asynchronously changing their values in $\{0,1\}$ according to a stochastic transition rule. The transition rule used by Hinton et. al. defines the probability of a unit to be in the 'on' state as the logistic function of the energy change resulting by changing the value of that unit. The BM can be described by an ergodic Markov chain in which the thermal equilibrium probability of attaining each state obeys the Boltzmann distribution which is a function of *only* the energy. By associating the set of possible propositions with subsets of network states, the probability of deciding upon each of these propositions can be measured by the probability of attaining the corresponding set of states. This probability is also affected by the temperature. As the temperature

This work was supported by the Weizmann Foundation for scientific research, by the University of California MICRO program, and by Bell Communications Research, Inc.

increases, the Boltzmann probability distribution become more uniform and thus the decision made is 'vague'. The lower the temperature the greater is the probability of attaining states with lower energy thereby leading to more 'distinctive' decisions.

This approach, while very attractive in principle, has two major drawbacks which make the complexity of the computations become non-feasible for nontrivial problems. The first is the need for thermal equilibrium in order to obtain the Boltzmann distribution. To make distinctive decisions a low temperature is required. This implies slower convergence towards thermal equilibrium. Generally, the method used to reach thermal equilibrium is simulated annealing (SA) (Kirkpatrick et. al., 1983) in which the temperature starts from a high value and is gradually reduced to the desired final value. In order to avoid 'freezing' of the network, the cooling schedule should be fairly slow. SA is thus time consuming and computationally expensive. The second drawback is due to the stochastic nature of the computation. Since the network state is a random vector, the desired probabilities have to be estimated by accumulating statistics of the network behavior for only a finite period of time. Hence, a trade-off between speed and accuracy is unavoidable.

In this paper, we propose a mechanism to alleviate the above computational drawbacks by converting the stochastic BM into a functionally equivalent deterministic network, which we call the Boltzmann Perceptron Network (BPN). The BPN circumvents the need for a Monte Carlo type of computation and instead evaluates the desired probabilities using a multilayer perceptron-like network. The very time consuming learning process for the BM is similarly replaced by a deterministic learning scheme, somewhat akin to the backpropagation algorithm, which is computationally affordable. The similarity between the learning algorithm of a BM having a layered structure and that of a two-layer perceptron has been recently pointed out by Hopfield (1987). In this paper we further elaborate on such an equivalence between the BM and the new perceptron-like network, and give the conditions under which the conversion of the stochastic BM into the deterministic BPN is possible. Unlike the original BM, the BPN is virtually always in thermal equilibrium and thus SA is no longer required. Nevertheless, the temperature still plays the same role and thus varying it may be beneficial to control the 'softness' of the decisions made by the BPN. Using the BPN as a soft classifier is described in details in (Yair and Gersho, 1989).

## THE BOLTZMANN PERCEPTRON NETWORK

Suppose we have a network of $K$ units connected through symmetric connection links with no self-feedback, so that the connection matrix $\Gamma$ is symmetric and zero-diagonal. Let us categorize the units into three different types: input, output and hidden units. The input pattern will be supplied to the network by clamping the input units, denoted by $\underline{x} = (x_1,..,x_i,..,x_I)^T$, with this pattern. $\underline{x}$ is a real-valued vector in $R^I$. The output of the network will be observed on the output units $\underline{y} = (y_1,..,y_m,..,y_M)^T$, which is a binary vector. The remaining units, denoted $\underline{v} = (v_1,..,v_j,..,v_J)^T$, are the hidden units, which are also binary-valued. The hidden and output units are asynchronously and randomly changing their binary values in {0,1} according to inputs they receive from other units.

The state of the network will be denoted by the vector $\underline{u}$ which is partitioned as follows: $\underline{u}^T = (\underline{x}^T, \underline{v}^T, \underline{y}^T)$. The energy associated with state $\underline{u}$ is denoted by $E_u$ and is given by:

$$-E_u = \tfrac{1}{2}\underline{u}^T\Gamma\underline{u} + \underline{u}^T\underline{\delta} \qquad (1)$$

where $\underline{\delta}$ is a vector of bias values, partitioned to comply with the partition of $\underline{u}$ as follows: $\underline{\delta}^T = (\underline{f}^T, \underline{c}^T, \underline{s}^T)$.

The transition from one state to another is performed by selecting each time one unit, say unit $k$, at random and determine its output value according to the following stochastic rule: set the output of the unit to 1 with probability $p_k$, and to 0 with a probability $1-p_k$. The parameter $p_k$ is determined locally by the $k$-th unit as a function of the energy change $\Delta E_k$ in the following fashion:

$$p_k = g(\Delta E_k) \quad ; \qquad\qquad g(x) \overset{\Delta}{=} \frac{1}{1+e^{-\beta x}} \quad . \tag{2}$$

$\Delta E_k = \{E_u(\text{unit } k \text{ is off}) - E_u(\text{unit } k \text{ is on})\}$, and $\beta = 1/T$ is a control parameter. $T$ is called the temperature and $g(\cdot)$ is the logistic function. With this transition rule the thermal equilibrium probability $P_u$ of attaining a state $\underline{u}$ obeys the Boltzmann distribution:

$$P_u = \frac{1}{Z_x} e^{-\beta E_u} \tag{3}$$

where $Z_x$, called the *partition function*, is a normalization factor (independent of $\underline{v}$ and $\underline{y}$) such that the sum of $P_u$ over all the $2^{J+M}$ possible states will sum to unity.

In order to use the network in a deterministic fashion rather than accumulate statistics while observing its random behavior, we should be able to *explicitly* compute the probability of attaining a certain vector $\underline{y}$ on the output units while $\underline{x}$ is clamped on the input units. This probability, denoted by $P_{y|x}$, can be expressed as:

$$P_{y|x} = \sum_{v \in B_J} P_{v,y|x} = \frac{1}{Z_x} \sum_{v \in B_J} e^{-\beta E_{v,y|x}} \tag{4}$$

where $B_J$ is the set of all binary vectors of length $J$, and $v, y|x$ denotes a state $\underline{u}$ in which a specific input vector $\underline{x}$ is clamped. The explicit evaluation of the desired probabilities therefore involves the computation of the partition function for which the number of operations grows exponentially with the number of units. That is, the complexity is $O(2^{J+M})$. Obviously, this is computationally unacceptable. Nevertheless, we shall see that under a certain restriction on the connection matrix $\Gamma$ the explicit computation of the desired probabilities becomes possible with a complexity of $O(JM)$ which is computationally feasible.

Let us assume that for each input pattern we have $M$ possible propositions which are associated with the $M$ output vectors: $I_M = \{\underline{I}_1, ... \underline{I}_m, ... \underline{I}_M\}$, where $\underline{I}_m$ is the $m$-th column of the $M \times M$ identity matrix. Any state of the network having output vector $\underline{y} = \underline{I}_m$ (for any $m$) will be denoted by $v, m|x$ and will be called a feasible state. All other state vectors $v, y|x$ for $\underline{y} \neq \underline{I}_m$ will be considered as intermediate steps between two feasible states. This redefinition of the network states is equivalent to redefining the states of the underlying Markov model of the network and thus conserves the equilibrium Boltzmann distribution. The probability of proposition $m$ for a given input $\underline{x}$, denoted by $P_{m|x}$, will be taken as the probability of obtaining output vector $\underline{y} = \underline{I}_m$ given that the output is one of the feasible values. That is,

$$P_{m|x} = Pr\{\underline{y} = \underline{I}_m \mid \underline{x}, \underline{y} \in I_M\} \tag{5}$$

which can be computed from (4) by restricting the state space to the $2^J M$ feasible state vectors and by setting $\underline{y} = \underline{I}_m$. The partition function, conditioned on restricting $\underline{y}$ to lie in the set of feasible outputs, $I_M$, is denoted by $\bar{Z}_x$ and is given by:

$$\bar{Z}_x = \sum_{y \in I_M} \sum_{v \in B_J} e^{-\beta E_{v,y|x}} \quad . \tag{6}$$

Let us now partition the connection matrix $\Gamma$ to comply with the partition of the state vector and rewrite the energy for the feasible state $v, m \mid x$ as:

$$-E_{v,m\mid x} = \underline{y}^T(R\underline{x}+Q\underline{1}_m+\tfrac{1}{2}D_2\underline{v}+\underline{c}) + \underline{1}_m^T(W\underline{x}+\tfrac{1}{2}D_3\underline{1}_m+\underline{s}) + \underline{x}^T(\tfrac{1}{2}D_1\underline{x}+\underline{f}) \quad . \qquad (7)$$

Since $\underline{x}$ is clamped on the input units, the last term in the energy expression serves only as a bias term for the energy which is independent of the binary units $\underline{v}$ and $\underline{y}$. Therefore, without any loss of generality it may be assumed that $D_1 = 0$ and $\underline{f} = \underline{0}$. The second term, denoted by $T_{m\mid x}$, can be simplified since $D_3$ has a zero diagonal. Hence,

$$T_{m\mid x} = \sum_{i=1}^{I} w_{mi} x_i + s_m \quad . \qquad (8)$$

The absence of the matrix $D_3$ in the energy expression means that interconnections between output units have no effect on the probability of attaining output vectors $\underline{y} \in I_M$, and may be assumed absent without any loss of generality.

Defining $L_m(\underline{x})$ to be:

$$L_m(\underline{x}) \overset{\Delta}{=} \beta T_{m\mid x} + \ln \left[ \sum_{\underline{v} \in B_J} e^{\beta \underline{v}^T(R\underline{x}+\underline{q}_m+\tfrac{1}{2}D_2\underline{v}+\underline{c})} \right] \quad , \qquad (9)$$

in which $\underline{q}_m$ is the $m$-th column of $Q$, the desired probabilities, $P_{m\mid x}$, for $m=1,..,M$ are obtained using (4) and (7) as a function of these quantities as follows:

$$P_{m\mid x} = \frac{1}{\bar{Z}_x} e^{L_m(\underline{x})} \qquad \text{with:} \qquad \bar{Z}_x = \sum_{m=1}^{M} e^{L_m(\underline{x})} \quad . \qquad (10)$$

The complexity of evaluating the desired probabilities $P_{m\mid x}$ is still exponential with the number of hidden units $J$ due to the sum in (9). However, if we impose the restriction that $D_2 = 0$, namely, the hidden units are not directly interconnected, then this sum becomes separable in the components $v_j$ and thus can be decomposed into the product of *only J* terms. This restricted connectivity of course imposes some restrictions on the capability of the network compared to that of a fully connected network. On the other hand, it allows the computation of the desired probabilities in a deterministic way with the attractive complexity of only $O(JM)$ operations. The tedious estimation of conditional expectations commonly required by the learning algorithm for a BM and the annealing procedure are avoided, and an accurate and computationally affordable scheme becomes possible. We thus suggest a trade-off in which the operation and learning of the BM are tremendously simplified and the *exact* decision probabilities are computed (rather than their statistical estimates) at the expense of a restricted connectivity, namely, no interconnections are allowed between the hidden units. Hence, in our scheme, the connection matrix, $\Gamma$, becomes zero block-diagonal, meaning that the network has connections only between units of different categories. This structure is shown schematically in Figure 1.

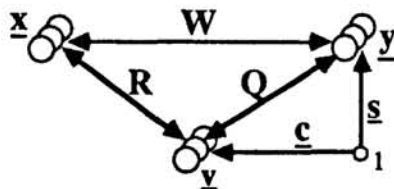

Figure 1.   Schematic architecture of the stochastic BM.

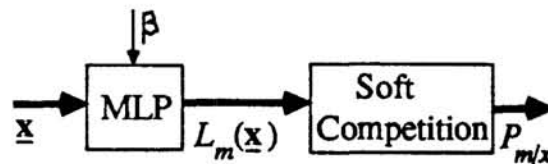

Figure 2.   Block diagram of the corresponding deterministic BPN.

By applying the property $D_2 = 0$ to (9), the sum over the space of hidden units, which can be explicitly written as the sum over all the $J$ components of $\underline{v}$, can be decomposed using the separability of the different $v_j$ components into a sum of only $J$ terms as follows:

$$L_m(\underline{x}) = \beta T_{m|x} + \sum_{j=1}^{J} f(V_j^{m|x}) \tag{11a}$$

where: $\qquad f(x) \triangleq \ln(1 + e^{\beta x}) \qquad$ and $\qquad V_j^{m|x} \triangleq \sum_{i=1}^{I} r_{ji}\, x_i + c_j + q_{jm} \tag{11b}$

$f(\cdot)$ is called the activation function. Note that as $\beta$ is increased $f(\cdot)$ approaches the linear threshold function in which a zero response is obtained for a negative input and a linear one (with slope $\beta$) for a positive input.

Finally, the desired probabilities $P_{m|x}$ can be expressed as a function of the $L_m(\underline{x})$ ($m = 1,..,M$) in an expression which can be regarded as the generalization of the logistic function to $M$ inputs:

$$P_{m|x} = \left[ 1 + \sum_{\substack{n=1 \\ n \neq m}}^{M} e^{-L_{m,n}(\underline{x})} \right]^{-1} \quad ; \qquad \text{where:} \qquad L_{n,m}(\underline{x}) = L_n(\underline{x}) - L_m(\underline{x}) \quad . \tag{12}$$

Eqs. (8) and (11) describe a two-layer feed-forward perceptron-like subnetwork which uses the nonlinearity $f(\cdot)$. It evaluates the quantity $L_m(\underline{x})$ which we call the *score* of the $m$-th proposition. Eq. (12) describes a competition between the scores $L_m(\underline{x})$ generated by the $M$ subnetworks ($m = 1,..,M$) which we call a *soft competition with lateral inhibition*. That is, If several scores receive relatively high values compared to the others, they will share, according to their relative strengths, the total amount (unity) of probability, while inhibiting the remaining probabilities to approach zero. For example, if one of the scores, say $L_k(\underline{x})$, is large compared to all the other scores, then the exponentiation of the pairwise score differences will result in $P_{k|x} \approx 1$ while the remaining probabilities will approach zero. Specifically, for any $n \neq k$, $P_{n|x} \approx \exp(-L_{k,n}(\underline{x}))$, which is essentially zero if $L_k(\underline{x})$ is sufficiently high. In other words, by being large compared to the others, $L_k(\underline{x})$ won the competition so that the corresponding probability $P_{k|x}$ approaches unity, while all the remaining probabilities have been attenuated by the high value of $L_k(\underline{x})$ to approach zero.

Let us examine the effect of the gain $\beta$ on this competition. When $\beta$ is increased, the slope of the activation function $f(\cdot)$ is increased thereby accentuating the differences between the $M$ contenders. In the limit when $\beta \rightarrow \infty$, *one* of the $L_m(\underline{x})$ will always be sufficiently large compared to the others, and thus *only one* proposition will win. The competition then becomes a winner-take-all competition. In this case, the network becomes a *maximum a posteriori* (MAP) decision scheme in which the $L_m(\underline{x})$ play the role of nonlinear discriminant functions and the most probable proposition for the given input pattern is chosen:

$$P_{k|x} = 1 \quad \text{for} \quad k = \underset{m}{\text{argmax}}\, \{ L_m(\underline{x}) \} \quad \text{and} \quad P_{n|x} = 0 \quad \text{for} \quad n \neq k \ . \tag{13}$$

This results coincides with our earlier observation that the temperature controls the 'softness' of the decision. The lower the temperature, the 'harder' is the competition and the more distinctive are the decisions. However, in contrast to the stochastic network, there is no need to gradually 'cool' the network to achieve a desired (low) temperature. Any desired value of $\beta$ is *directly* applicable in the BPN scheme. The above notion of soft competition has its merits in a wider scope of applications apart from its role in the BPN classifier. In many competitive schemes a soft competition between a set of contenders has

a substantial benefit in comparison to the winner-take-all paradigm. The above competition scheme which can be implemented by a two-layer feed-forward network thus offers a valuable scheme for such purposes.

The block diagram of the BPN is depicted in Figure 2. The BPN is thus a four-layer feed-forward deterministic network. It is comprised of a two-layer perceptron-like network followed by a two-layer competition network. The competition can be 'hard' (winner-take-all) or 'soft' (graded decision) and is governed by a single gain parameter $\beta$.

## THE LEARNING ALGORITHM

Let us denote the BPN output by the $M$-dimensional probability vector $\underline{P}_x$, where: $\underline{P}_x = (P_{1|x},...,P_{m|x},...,P_{M|x})^T$. For any given set of weights $\underline{\theta}$, the BPN realizes some deterministic mapping $\Psi: R^I \rightarrow [0,1]^M$ so that $\underline{P}_x = \Psi(\underline{x})$. The objective of learning is to determine the mapping $\Psi$ (by estimating the set of parameters $\underline{\theta}$) which 'best' explains a finite set of examples given from the desired mapping and are called the *training set*. The training set is specified by a set of $N$ patterns $\{\underline{x}_1,...,\underline{x}_n,...\underline{x}_N\}$ (denoted for simplicity by $\{\underline{x}\}$), the *a priori* probability for each training pattern $\underline{x}$: $Q(\underline{x})$, and the desired mapping for each pattern $\underline{x}$: $\underline{Q}_x = (Q_{1|x},...,Q_{m|x},...,Q_{M|x})^T$, where $Q_{m|x} = Pr\{$proposition $m \mid \underline{x}\}$ is the desired probability. For each input pattern presented to the BPN, the actual output probability vector $\underline{P}_x$ is, in general, different from the desired one, $\underline{Q}_x$. We denote by $G_x$ the distortion between the actual $\underline{P}_x$ and the desired $\underline{Q}_x$ for an input pattern $\underline{x}$. Thus, our task is to determine the network weights (and the bias values) $\underline{\theta}$ so that, on the average, the distortion over the whole set of training patterns will be minimized. Adopting the original distortion measure suggested for Boltzmann machines, the average distortion, $G(\underline{\theta})$, is given by:

$$G(\underline{\theta}) = \sum_{\underline{x}} Q(\underline{x}) G_x(\underline{\theta}) \quad ; \quad G_x(\underline{\theta}) = \sum_{m=1}^{M} Q_{m|x} \ln[Q_{m|x} / P_{m|x}(\underline{\theta})] \tag{14}$$

which is always non-negative since $\underline{P}_x$ and $\underline{Q}_x$ are probability vectors. To minimize $G(\underline{\theta})$ a gradient based minimization search is used. Specifically, a Partial Conjugate Gradient (PCG) search (Fletcher and Reeves, 1964; Luenberger, 1984) was found to be significantly more efficient than the ordinary steepest descent approach which is so widely used in multilayer perceptrons. A further discussion supporting this finding is given in (Yair and Gersho, 1989). For each set of weights we thus have to be able to compute the gradient $\underline{g} = \nabla_\theta G$ of the cost function $G(\underline{\theta})$. Let us denote the components of the 'instantaneous' gradient by $G_m^{s|x} = \partial G_x / \partial s_m$, $G_{jm}^{q|x} = \partial G_x / \partial q_{jm}$, $G_j^{c|x} = \partial G_x / \partial c_j$, $G_{mi}^{w|x} = \partial G_x / \partial w_{mi}$, $G_{ji}^{r|x} = \partial G_x / \partial r_{ji}$. To get the full gradient, the instantaneous components should be accumulated while the input patterns are presented (one at a time) to the network, until one full cycle through the whole training set is completed.

It is straightforward to show that the gradient may be evaluated in a recursive manner, in a fashion somewhat similar to the evaluation of the gradient by the backpropagation algorithm used for feed-forward networks (Rumelhart et. al., 1986). The evaluation of the gradient is accomplished by propagating the errors $e_{m|x} = Q_{m|x} - P_{m|x}$ through a linear network, termed the Error Propagation Network (EPN), as follows:

$$G_m^{s|x} = -\beta e_{m|x} \quad ; \quad G_{mi}^{w|x} = x_i G_m^{s|x}$$

$$G_{jm}^{q|x} = b_j^{m|x} G_m^{s|x} \quad ; \quad G_j^{c|x} = \sum_{m=1}^{M} G_{jm}^{q|x} \quad ; \quad G_{ji}^{r|x} = x_i G_j^{c|x} \quad . \tag{15}$$

The only new variables required are the $b_j^{m|x}$, given by $b_j^{m|x} = g(V_j^{m|x})$, which can be

easily obtained by applying the logistic nonlinearity to $V_j^{m|x}$. The above error propagation scheme can also be written in a matrix form. Let us define the following notation: $\underline{e}_x = (e_{1|x},...,e_{m|x},...,e_{M|x})^T$. $\mathbf{E}_x$ will be a diagonal $M \times M$ matrix whose diagonal is $\underline{e}_x$. $\mathbf{B}_x = [b_j^{m|x}]$, a $J \times M$ matrix. Let $\underline{1}_M$ denote a column vector of length $M$ whose components are all 1's. Similarly we will define the vectors: $\underline{G}^{\lambda|x} = (..,G_\varepsilon^{\lambda|x},..)^T$, and the matrices: $\mathbf{G}^{\lambda|x} = [G_{\varepsilon\eta}^{\lambda|x}]$ with the appropriate dimensions (for any $\lambda$, $\varepsilon$ and $\eta$). Hence, the error propagation can be written as:

$$\underline{G}^{s|x} = -\beta \underline{e}_x \quad ; \quad \mathbf{G}^{w|x} = \underline{G}^{s|x} \mathbf{x}^T$$
$$\mathbf{G}^{q|x} = -\beta \mathbf{B}_x \mathbf{E}_x \quad ; \quad \underline{G}^{c|x} = \mathbf{G}^{q|x} \underline{1}_M \quad ; \quad \mathbf{G}^{r|x} = \underline{G}^{c|x} \underline{\mathbf{x}}^T \quad . \tag{16}$$

The EPN is depicted in Figure 3. This is a linear system composed of inner and outer products between matrices, which can be efficiently implemented using a neural network. The gradient **g** is used in the PCG update formula in which a new set of weights is created and is used for the next update iteration. The learning scheme of the BPN is given in Figure 4.

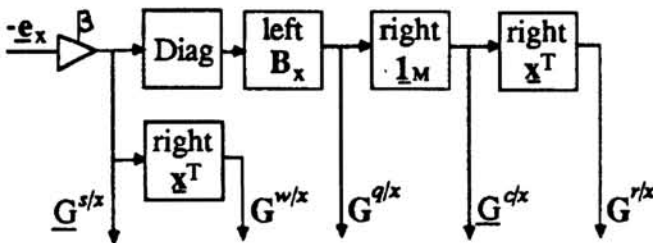

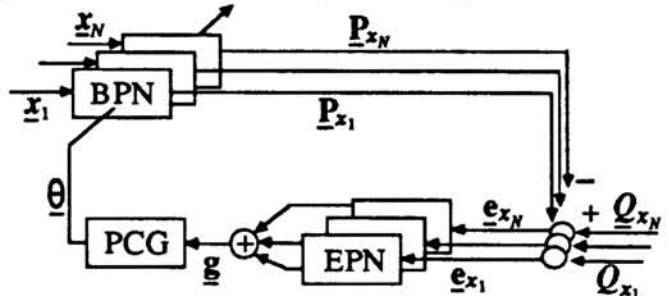

Figure 3. The Error Propagation Network (EPN). 'Diag' is a diagonalization operator. 'right' and 'left' are right and left multipliers, respectively.

Figure 4. The learning scheme. The BPN outputs, $\underline{P}_x$, are compared with the desired probabilities, $\underline{Q}_x$. The resulted errors, $\underline{e}_x$, propagate through the EPN to form the gradient **g** which is used in the PCG alg. to create the new weights.

## SIMULATION RESULTS

We now present several simulation results of two-class classification problems with Gaussian sources. That is, we have two propositions represented by class 0 or class 1. Suppose there are $L$ random sources ($i=1,..,L$) over the input space, some of them are attributed to class 0, and the others to class 1. Each time, a source is chosen according to an *a priori* probability $P(i)$. When chosen, the $i$-th source then emits a pattern $\underline{x}$ according to a probability density $Q_{x|i}(\underline{x})$. Measuring a pattern $\underline{x}$ it is desired to decide upon the most probable origin class - in the binary decision problem (MAP classifier), or obtain some estimate to $Q_{1|x}$, the probability that class 1 emitted this pattern - in the soft classification problem. In the learning phase, a training set of size $N$ was presented to the network, and the weights were iteratively modified by the learning scheme (Figure 4) until convergence. The final weights were used to construct a BPN classifier, which was then tested for new input patterns. The output classification probability of the BPN, $P_{1|x}(\underline{x})$, was compared with the true (Bayesian) conditional probability, $Q_{1|x}(\underline{x})$ which was computed analytically. Results are shown in Figures 5-7. In Figure 5, two symmetric equi-probable Gaussian sources with substantial overlap were used, one for each class. The network was trained on $N=8$ patterns with gain $\beta=1$. Figure 5b shows how the BPN performs for the problem given in Figure 5a. For $\beta=1$, i.e., when the same gain is used in both the training and classification phases, there is an almost perfect match between the BPN output, $P_{1|x}(x)$, denoted in the figures by '$\beta=1$', and the true curve, $Q_{1|x}(x)$. For $\beta=10$, the high gain winner-take-all

competition is taking place and the classifier becomes, practically, a binary (yes/no) decision network. In Fig. 6 disconnected decision regions were formed by using four sources, two of which were attributed to each class. Again, a nearly perfect match can be seen between actual ($\beta = 1$) and desired ($Q_{1|x}$) outputs. Also, the simplicity of making 'hard' classification decisions by increasing the gain is again illustrated ($\beta = 10$). In Fig. 7 the classifier was required to find the boundary (expressed by $Q_{1|x} = 0.5$) between two 2D classes.

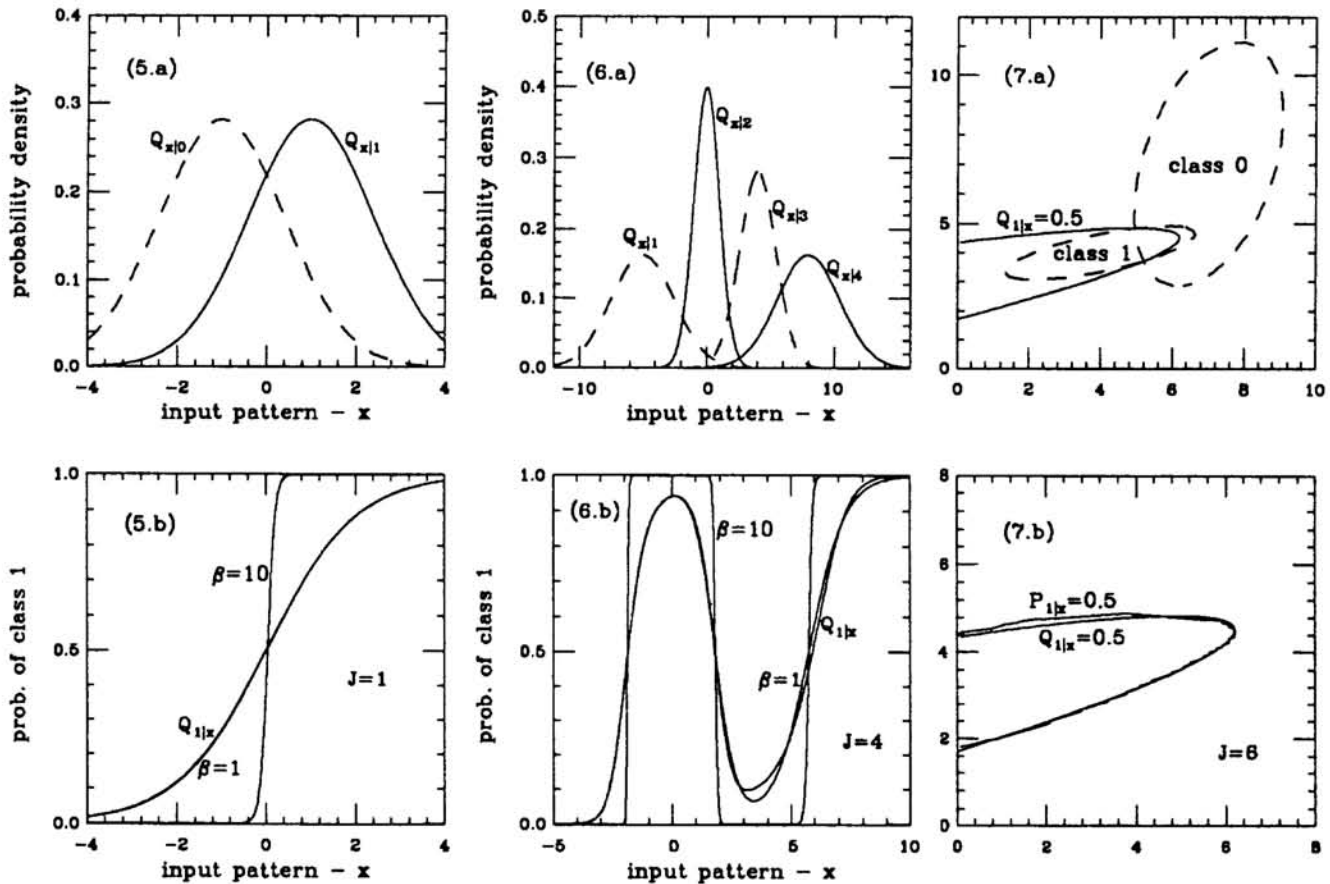

**Figure 5.**          **Figure 6.**          **Figure 7.**

**Figure 5:** Classification for Gaussian sources. (5.a) The two sources. (5.b) 'Soft' ($\beta = 1$) and 'hard' ($\beta = 10$) classifications versus $Q_{1|x}$. $J$ indicates the number of hidden units used.

**Figure 6:** Classification for disconnected decision regions. (6.a) The sources used: dashed lines indicate class 0 and solid lines - class 1. (6.b) Soft ($\beta = 1$) and hard ($\beta = 10$) classifications versus $Q_{1|x}$.

**Figure 7:** Classification in a 2D space. (7.a) The two classes and the true boundary indicated by $Q_{1|x} = 0.5$. (7.b) The boundary found by the BPN, marked by $P_{1|x} = 0.5$, versus the true one.

## References

Fletcher, R., Reeves, C.M. (1964). Function minimization by conjugate gradients. *Computer J.*, **7**, 149-154.

Hinton, G.E., Sejnowski T.R., & Ackley D.H. (1984). Boltzmann machines: constraint satisfaction networks that learn. *Carnegie-Melon Technical Report*, CMU-CS-84-119.

Hopfield, J.J. (1987). Learning algorithms and probability distributions in feed-forward and feed-back networks. *Proc. Natl. Acad. Sci. USA*, **84**, 8429-8433.

Kirkpatrick, S., Gelatt, C.D., & Vecchi M.P. (1983). Optimization by simulated annealing. *Science*, **220**, 671-680.

Luenberger, D.G. (1984). *Linear and nonlinear programming*, Addison-Wesley, Reading, Mass.

Rumelhart, D.E., Hinton, G.E., & Williams R.J. (1986). Learning internal representations by error propagation. In D.E. Rumelhart & J.L. McClelland (Eds.), *Parallel Distributed Processing.*, MIT Press/Bradford Books.

Yair, E., & Gersho, A. (1989). The Boltzmann perecptron network: a soft classifier. Submitted to the *Journal of Neural Networks*, December, 1988.
